# On the Computational Complexity of Networks of Spiking Neurons

## (Extended Abstract)

**Wolfgang Maass**
Institute for Theoretical Computer Science
Technische Universitaet Graz
A-8010 Graz, Austria
e-mail: maass@igi.tu-graz.ac.at

## Abstract

We investigate the computational power of a formal model for networks of spiking neurons, both for the assumption of an unlimited timing precision, and for the case of a limited timing precision. We also prove upper and lower bounds for the number of examples that are needed to train such networks.

## 1 Introduction and Basic Definitions

There exists substantial evidence that timing phenomena such as temporal differences between spikes and frequencies of oscillating subsystems are integral parts of various information processing mechanisms in biological neural systems (for a survey and references see e.g. Abeles, 1991; Churchland and Sejnowski, 1992; Aertsen, 1993). Furthermore simulations of a variety of specific mathematical models for networks of spiking neurons have shown that temporal coding offers interesting possibilities for solving classical benchmark-problems such as associative memory, binding, and pattern segmentation (for an overview see Gerstner et al., 1992). Some aspects of these models have also been studied analytically, but almost nothing is known about their computational complexity (see Judd and Aihara, 1993, for some first results in this direction). In this article we introduce a simple formal model SNN for networks of spiking neurons that allows us to model the most important timing phenomena of neural nets (including synaptic modulation), and we prove upper and lower bounds for its computational power and learning complexity. Further

details to the results reported in this article may be found in Maass, 1994a,1994b, 1994c.

**Definition of a Spiking Neuron Network (SNN):**  *An SNN $\mathcal{N}$ consists of*

- *a finite directed graph $\langle V, E \rangle$ (we refer to the elements of V as "neurons" and to the elements of E as "synapses")*
- *a subset $V_{in} \subseteq V$ of input neurons*
- *a subset $V_{out} \subseteq V$ of output neurons*
- *for each neuron $v \in V - V_{in}$ a threshold-function $\Theta_v : \mathbf{R}^+ \to \mathbf{R} \cup \{\infty\}$ (where $\mathbf{R}^+ := \{x \in \mathbf{R} : x \geq 0\}$)*
- *for each synapse $\langle u, v \rangle \in E$ a response-function $\varepsilon_{u,v} : \mathbf{R}^+ \to \mathbf{R}$ and a weight- function $w_{u,v} : \mathbf{R}^+ \to \mathbf{R}$ .*

*We assume that the firing of the input neurons $v \in V_{in}$ is determined from outside of $\mathcal{N}$, i.e. the sets $F_v \subseteq \mathbf{R}^+$ of firing times ("spike trains") for the neurons $v \in V_{in}$ are given as the input of $\mathcal{N}$. Furthermore we assume that a set $T \subseteq \mathbf{R}^+$ of potential firing times has been fixed.*

*For a neuron $v \in V - V_{in}$ one defines its set $F_v$ of firing times recursively. The first element of $F_v$ is $\inf\{t \in T : P_v(t) \geq \Theta_v(0)\}$ , and for any $s \in F_v$ the next larger element of $F_v$ is $\inf\{t \in T : t > s \text{ and } P_v(t) \geq \Theta_v(t - s)\}$ , where the potential function $P_v : \mathbf{R}^+ \to \mathbf{R}$ is defined by*

$$P_v(t) := 0 + \sum_{u \,:\, \langle u, v \rangle \in E} \sum_{s \in F_u \,:\, s < t} w_{u,v}(s) \cdot \varepsilon_{u,v}(t - s) \ .$$

*The firing times ("spike trains") $F_v$ of the output neurons $v \in V_{out}$ that result in this way are interpreted as the output of $\mathcal{N}$.*

Regarding the set $T$ of potential firing times we consider in this article the case $T = \mathbf{R}^+$ (SNN with continuous time) and the case $T = \{i \cdot \mu : i \in \mathbf{N}\}$ for some $\mu$ with $1/\mu \in \mathbf{N}$ (SNN with discrete time).

We assume that for each SNN $\mathcal{N}$ there exists a bound $\tau_{\mathcal{N}} \in \mathbf{R}$ with $\tau_{\mathcal{N}} > 0$ such that $\Theta_v(x) = \infty$ for all $x \in (0, \tau_{\mathcal{N}})$ and all $v \in V - V_{in}$ ($\tau_{\mathcal{N}}$ may be interpreted as the minimum of all "refractory periods" $\tau_{ref}$ of neurons in $\mathcal{N}$). Furthermore we assume that all "input spike trains" $F_v$ with $v \in V_{in}$ satisfy $|F_v \cap [0, t]| < \infty$ for all $t \in \mathbf{R}^+$. On the basis of these assumptions one can also in the continuous case easily show that the firing times are well-defined for all $v \in V - V_{in}$ (and occur in distances of at least $\tau_{\mathcal{N}}$).

**Input- and Output-Conventions:** For simulations between SNN's and Turing machines we assume that the SNN either gets an input (or produces an output) from $\{0, 1\}^*$ in the form of a spike-train (i.e. one bit per unit of time), or encoded into the phase-difference of just two spikes. *Real-valued* input or output for an SNN is always encoded into the phase-difference of two spikes.

**Remarks**
**a)** In models for *biological neural systems* one assumes that if $x$ time-units have

passed since its last firing, the current threshold $\Theta_v(x)$ of a neuron $v$ is "infinite" for $x < \tau_{ref}$ (where $\tau_{ref}$ = refractory period of neuron $v$), and then approaches quite rapidly from above some constant value. A neuron $v$ "fires" (i.e. it sends an "action potential" or "spike" along its axon) when its current membrane potential $P_v(t)$ at the axon hillock exceeds its current threshold $\Theta_v$. $P_v(t)$ is the sum of various postsynaptic potentials $w_{u,v}(s) \cdot \varepsilon_{u,v}(t-s)$. Each of these terms describes an *excitatory* (EPSP) or *inhibitory* (IPSP) *postsynaptic potential* at the axon hillock of neuron $v$ at time $t$, as a result of a spike that had been generated by a "presynaptic" neuron $u$ at time $s$, and which has been transmitted through a synapse between both neurons. Recordings of an EPSP typically show a function that has a constant value $c$ ($c$ = resting membrane potential; e.g. $c = -70mV$) for some initial time-interval (reflecting the axonal and synaptic transmission time), then rises to a peak-value, and finally drops back to the same constant value $c$. An IPSP tends to have the negative shape of an EPSP. For the sake of mathematical simplicity we assume in the SNN-model that the constant initial and final value of all response-functions $\varepsilon_{u,v}$ is equal to 0 (in other words: $\varepsilon_{u,v}$ models the *difference* between a postsynaptic potential and the resting membrane potential $c$). Different presynaptic neurons $u$ generate postsynaptic potentials of different sizes at the axon hillock of a neuron $v$, depending on the size, location and current state of the synapse (or synapses) between $u$ and $v$. This effect is modelled by the weight-factors $w_{u,v}(s)$.

The precise shapes of threshold-, response-, and weight-functions vary among different biological neural systems, and even within the same system. Fortunately one can prove significant *upper bounds* for the computational complexity of SNN's $\mathcal{N}$ without any assumptions about the *specific shapes* of these functions of $\mathcal{N}$. Instead, we only assume that they are of a reasonably simple *mathematical structure*.

**b)** In order to prove *lower bounds* for the computational complexity of an SNN $\mathcal{N}$ one is forced to make more specific assumptions about these functions. All lower bound results that are reported in this article require only some rather weak *basic assumptions* about the response- and threshold-functions. They mainly require that EPSP's have some (arbitrarily short) segment where they increase linearly, and some (arbitrarily short) segment where they decrease linearly (for details see Maass, 1994a, 1994b).

**c)** Although the model SNN is apparently more "realistic" than all models for biological neural nets whose computational complexity has previously been analyzed, it deliberately sacrifices a large number of more intricate biological details for the sake of mathematical tractability. Our model is closely related to those of (Buhmann and Schulten, 1986), and (Gerstner, 1991, 1992). Similarly as in (Buhmann and Schulten, 1986) we consider here only the deterministic case.

**d)** The model SNN is also suitable for investigating algorithms that involve *synaptic modulation* at various time-scales. Hence one can investigate within this framework not only the complexity of algorithms for supervised and unsupervised learning, but also the potential *computational* power of rapid weight-changes *within* the course of a computation. In the theorems of this paper we allow that the value of a weight $w_{u,v}(s)$ at a firing time $s \in F_u$ is defined by an *algebraic computation tree* (see van Leeuwen, 1990) in terms of its value at previous firing times $s' \in F_u$ with $s' < s$, some preceding firing times $\tilde{s} < s$ of arbitrary other neurons, and arbitrary real-valued parameters. In this way $w_{u,v}(s)$ can be defined by different rational functions

of the abovementioned arguments, depending on the numerical relationship between these arguments (which can be evaluated by comparing first the relative size of arbitrary rational functions of these arguments). As a simple special case one can for example increase $w_{u,v}$ (perhaps up to some specified saturation-value) as long as neurons $u$ and $v$ fire coherently, and decrease $w_{u,v}$ otherwise.

For the sake of simplicity in the statements of our results we assume in this extended abstract that the algebraic computation tree for each weight $w_{u,v}$ involves only $O(1)$ tests and rational functions of degree $O(1)$ that depend only on $O(1)$ of the abovementioned arguments. Furthermore we assume in Theorems 3, 4 and 5 that either each weight is an arbitrary time-invariant real, or that each current weight is rounded off to bit-length poly$(\log p_N)$ in binary representation, and does not depend on the times of firings that occured longer than time $O(1)$ ago. Furthermore we assume in Theorems 3 and 5 that the parameters in the algebraic computation tree are rationals of bit-length $O(\log p_N)$.

e) It is well-known that the *Vapnik-Chervonenkis dimension ("VC-dimension")* of a neural net $\mathcal{N}$ (and the *pseudo-dimension* for the case of a neural net $\mathcal{N}$ with *real-valued* output, with some suitable fixed norm for measuring the error) can be used to bound the number of examples that are needed to train $\mathcal{N}$ (see Haussler, 1992). Obviously these notions have to be *defined differently* for a network with *time-dependent* weights. We propose to define the VC-dimension (pseudo-dimension)of an SNN $\mathcal{N}$ with time-dependent weights as the VC-dimension (pseudo-dimension) of the class of all functions that can be computed by $\mathcal{N}$ with different assignments of values to the real-valued (or rational-valued) parameters of $\mathcal{N}$ that are involved in the definitions of the piecewise rational response-, threshold-, and weight-functions of $\mathcal{N}$. In a biological neural system $\mathcal{N}$ these parameters might for example reflect the concentrations of certain chemical substances that are known to modulate the behavior of $\mathcal{N}$.

f) The focus in the investigation of computations in biological neural systems differs in two essential aspects from that of classical computational complexity theory. First, one is not only interested in single computations of a neural net for unrelated inputs $x$, but also in its ability to process an interrelated sequence $(\langle x(i), y(i) \rangle)_{i \in \mathbf{N}}$ of inputs and outputs, which may for example include an initial training sequence for learning or associative memory. Secondly, exact timing of computations is all-important in biological neural nets, and many tasks have to be solved within a specific number of steps. Therefore an analysis in terms of the notion of a *real-time computation* and *real-time simulation* appears to be more adequate for models of biological neural nets than the more traditional analysis via complexity classes.

One says that a sequence $(\langle x(i), y(i) \rangle)_{i \in \mathbf{N}}$ is *processed in real-time* by a machine $M$, if for every $i \in \mathbf{N}$ the machine $M$ outputs $y(i)$ within a constant number $c$ of computation steps after having received input $x(i)$. One says that $M'$ *simulates $M$ in real-time* (with delay factor $\Delta$), if every sequence that is processed in real-time by $M$ (with some constant $c$), can also be processed in real-time by $M'$ (with a constant $\Delta \cdot c$). For SNN's $M$ we count each spike in $M$ as a computation step.

These definitions imply that a real-time simulation of $M$ by $M'$ is a special case of a linear-time simulation, and hence that any problem that can be solved by $M$ with a certain time complexity $t(n)$, can be solved by $M'$ with time complexity $O(t(n))$

(see Maass, 1994a, 1994b, for details).

## 2 Networks of Spiking Neurons with Continuous Time

**Theorem 1:** *If the response- and threshold-functions of the neurons satisfy some rather weak basic assumptions (see Maass, 1994a, 1994b), then one can build from such neurons for any given $d \in N$ an SNN $\mathcal{N}_{TM}(d)$ of finite size with rational delays that can simulate with a suitable assignment of rational values from $[0, 1]$ to its weights any Turing machine with at most d tapes in real-time.*

*Furthermore $\mathcal{N}_{TM}(2)$ can compute any function $F : \{0,1\}^* \to \{0,1\}^*$ with a suitable assignment of real values from $\overline{[0, 1]}$ to its weights.*

The fixed SNN $\mathcal{N}_{TM}(d)$ of Theorem 1 can simulate Turing machines whose tape content is much larger than the size of $\mathcal{N}_{TM}(d)$, by encoding such tape content into the phase-difference between two oscillators. The proof of Theorem 1 transforms arbitrary computations of Turing machines into operations on such phase-differences.

The last part of Theorem 1 implies that the VC-dimension of some finite SNN's is infinite. In contrast to that the following result shows that one can give finite bounds for the VC-dimension of those SNN's that only use a bounded numbers of spikes in their computation. Furthermore the last part of the claim of Theorem 2 implies that their VC-dimension may in fact grow linearly with the number $S$ of spikes that occur in a computation.

**Theorem 2:** *The VC-dimension and pseudo-dimension of any SNN $\mathcal{N}$ with piecewise linear response- and threshold-functions, arbitrary real-valued parameters and time-dependent weights (as specified in section 1) can be bounded (even for real-valued inputs and outputs) by $O(|E| \cdot |V| \cdot S(\log |V| + \log S))$ if $\mathcal{N}$ uses in each computation at most $S$ spikes.*

*Furthermore one can construct SNN's (with any response- and threshold-functions that satisfy our basic assumptions, with fixed rational parameters and rational time-invariant weights) whose VC-dimension is for computations with up to $S$ spikes as large as $\Omega(|E| \cdot S)$.*

We refer to Maass, 1994a, 1994c, for *upper bounds* on the computational power of SNN's with continuous time.

## 3 Networks of Spiking Neurons with Discrete Time

In this section we consider the case where all firing times of neurons in $\mathcal{N}$ are multiples of some $\mu$ with $1/\mu \in \mathbf{N}$. We restrict our attention to the biologically plausible case where there exists some $t_{\mathcal{N}} \geq 1$ such that for all $x > t_{\mathcal{N}}$ all response functions $\varepsilon_{u,v}(x)$ have the value 0 and all threshold functions $\Theta_v(x)$ have some arbitrary constant value. If $t_{\mathcal{N}}$ is chosen minimal with this property, we refer to $p_{\mathcal{N}} := \lceil t_{\mathcal{N}}/\mu \rceil$ as the *timing-precision of $\mathcal{N}$*. Obviously for $p_{\mathcal{N}} = 1$ the SNN is equivalent to a "non-spiking" neural net that consists of linear threshold gates, whereas a SNN with continuous time may be viewed as the opposite extremal case for $p_{\mathcal{N}} \to \infty$.

The following result provides a significant upper bound for the computational power of an SNN with discrete time, even in the presence of arbitrary real-valued parameters and weights. Its proof is technically rather involved.

**Theorem 3:** *Assume that $\mathcal{N}$ is an SNN with timing-precision $p_\mathcal{N}$, piecewise polynomial response- and piecewise rational threshold-functions with arbitrary real-valued parameters, and weight-functions as specified in section 1.*

*Then one can simulate $\mathcal{N}$ for boolean valued inputs in real-time by a Turing machine with $poly(|V|, \log p_\mathcal{N}, \log 1/\tau_\mathcal{N})$ states and $poly(|V|, \log p_\mathcal{N}, t_\mathcal{N}/\tau_\mathcal{N})$ tape-cells.*

*On the other hand any Turing machine with $q$ states that uses at most $s$ tape-cells can be simulated in real-time by an SNN $\mathcal{N}$ with any response- and threshold-functions that satisfy our basic assumptions, with rational parameters and time-invariant rational weights, with $O(q)$ neurons, $\log p_\mathcal{N} = O(s)$, and $t_\mathcal{N}/\tau_\mathcal{N} = O(1)$.*

The next result shows that the VC-dimension of any SNN with discrete time is finite, and grows proportionally to $\log p_\mathcal{N}$. The proof of its lower bound combines a new explicit construction with that of Maass, 1993.

**Theorem 4:** *Assume that the SNN $\mathcal{N}$ has the same properties as in Theorem 3. Then the VC-dimension and the pseudo-dimension of $\mathcal{N}$ (for arbitrary real valued inputs) can be bounded by $O(|E| \cdot |V| \cdot \log p_\mathcal{N})$, independently of the number of spikes in its computations.*

*Furthermore one can construct SNN's $\mathcal{N}$ of this type with any response- and threshold-functions that satisfy our basic assumptions, with rational parameters and time-invariant rational weights, so that $\mathcal{N}$ has (already for boolean inputs) a VC-dimension of at least $\Omega(|E|(\log p_\mathcal{N} + \log |E|))$.*

## 4   Relationships to other Computational Models

We consider here the relationship between SNN's with discrete time and recurrent *analog neural nets*. In the latter no "spikes" or other non-trivial timing-phenomena occur, but the output of a gate consists of the "analog" value of some squashing- or activation function that is applied to the weighted sum of its inputs. See e.g. (Siegelmann and Sontag, 1992) or (Maass, 1993) for recent results about the computational power of such models. We consider in this section a perhaps more "realistic" version of such models $\mathcal{N}$, where the output of each gate is rounded off to an integer multiple of some $\frac{1}{a}$ (with $a \in \mathbf{N}$). We refer to $a$ as the *number of activation levels of $\mathcal{N}$*.

It is an interesting open problem whether such analog neural nets (with gate-outputs interpreted as firing rates) or networks of spiking neurons provide a more adequate computational model for biological neural systems. Theorem 5 shows that in spite of their quite different structure the computational power of these two models is in fact closely related.

On the side the following theorem also exhibits a new subclass of deterministic finite automata (DFA's) which turns out to be of particular interest in the context of neural nets. We say that a DFA $M$ is a *sparse DFA of size $s$* if $M$ can be realized by a Turing machine with $s$ states and space-bound $s$ (such that each step of $M$ corresponds to one step of the Turing machine). Note that a sparse DFA may have exponentially in $s$ many states, but that only $poly(s)$ bits are needed to describe its

transition function. Sparse DFA's are relatively easy to construct, and hence are very useful for demonstrating (via Theorem 5) that a specific task can be carried out on a "spiking" neural net with a realistic timing precision (respectively on an analog neural net with a realistic number of activation levels).

**Theorem 5:** *The following classes of machines have closely related computational power in the sense that there is a polynomial p such that each computational model from any of these classes can be simulated in real-time (with delay-factor $\leq p(s)$) by some computational model from any other class (with the size-parameter s replaced by p(s)):*

- *sparse DFA's of size s*
- *SNN's with $O(1)$ neurons and timing precision $2^s$*
- *recurrent analog neural nets that consist of $O(1)$ gates with piecewise rational activation functions with $2^s$ activation levels, and parameters and weights of bit-length $\leq s$*
- *neural nets that consist of s linear threshold gates (with recurrencies) with arbitrary real weights.*

The result of Theorem 5 is remarkably stable since it holds no matter whether one considers just SNN's $\mathcal{N}$ with $O(1)$ neurons that employ very simple fixed piecewise linear response- and threshold-functions with parameters of bit-length $O(1)$ (with $t_{\mathcal{N}}/\tau_{\mathcal{N}} = O(1)$ and time-invariant weights of bit-length $\leq s$), or if one considers SNN's $\mathcal{N}$ with $s$ neurons with arbitrary piecewise polynomial response- and piecewise rational threshold-functions with arbitrary real-valued parameters, $t_{\mathcal{N}}/\tau_{\mathcal{N}} \leq s$, and time-dependent weights (as specified in section 1).

# 5  Conclusion

We have introduced a simple formal model SNN for networks of spiking neurons, and have shown that significant bounds for its computational power and sample complexity can be derived from rather weak assumptions about the mathematical structure of its response-, threshold-, and weight-functions. Furthermore we have established quantitative relationships between the computational power of a model for networks of spiking neurons with a limited timing precision (i.e. SNN's with discrete time) and a quite realistic version of recurrent analog neural nets (with a bounded number of activation levels). The simulations which provide the proof of this result create an interesting link between computations with spike-coding (in an SNN) and computations with frequency-coding (in analog neural nets). We also have established such relationships for the case of SNN's with continuous time (see Maass 1994a, 1994b, 1994c), but space does not permit to report these results in this article.

The Theorems 1 and 5 of this article establish the existence of mechanisms for simulating arbitrary Turing machines (and hence *any* common computational model) on an SNN. As a consequence one can now demonstrate that a concrete task (such as binding, pattern-matching, associative memory) can be carried out on an SNN by simply showing that some arbitrary common computational model can carry out that task. Furthermore one can bound the required *timing-precision* of the SNN in terms of the *space* needed on a Turing machine.

Since we have based our investigations on the rather refined notion of a *real-time* simulation, our results provide information not only about the possibility to implement computations, but also *adaptive behavior* on networks of spiking neurons.

**Acknowledgement**

I would like to thank Wulfram Gerstner for helpful discussions.

# References

M. Abeles. (1991) Corticonics: Neural Circuits of the Cerebral Cortex. *Cambridge University Press.*

A. Aertsen. ed. (1993) Brain Theory: Spatio-Temporal Aspects of Brain Function. *Elsevier.*

J. Buhmann, K. Schulten. (1986) Associative recognition and storage in a model network of physiological neurons. *Biol. Cybern.* 54: 319-335.

P. S. Churchland, T. J. Sejnowski. (1992) The Computational Brain. *MIT-Press.*

W. Gerstner. (1991) Associative memory in a network of "biological" neurons. *Advances in Neural Information Processing Systems, vol. 3, Morgan Kaufmann:* 84-90.

W. Gerstner, R. Ritz, J. L. van Hemmen. (1992) A biologically motivated and analytically soluble model of collective oscillations in the cortex. *Biol. Cybern.* 68: 363-374.

D. Haussler. (1992) Decision theoretic generalizations of the PAC model for neural nets and other learning applications. *Inf. and Comput.* 95: 129-161.

K. T. Judd, K. Aihara. (1993) Pulse propagation networks: A neural network model that uses temporal coding by action potentials. *Neural Networks* 6: 203-215.

J. van Leeuwen, ed. (1990) Handbook of Theoretical Computer Science, vol. A: Algorithms and Complexity. *MIT-Press.*

W. Maass. (1993) Bounds for the computational power and learning complexity of analog neural nets. *Proc. 25th Annual ACM Symposium on the Theory of Computing,* 335-344.

W. Maass. (1994a) On the computational complexity of networks of spiking neurons (extended abstract). *TR 393 from May 1994 of the Institutes for Information Processing Graz* (for a more detailed version see the file *maass.spiking.ps.Z* in the *neuroprose archive*).

W. Maass. (1994b) Lower bounds for the computational power of networks of spiking neurons. *Neural Computation,* to appear.

W. Maass. (1994c) Analog computations on networks of spiking neurons (extended abstract). Submitted for publication.

H. T. Siegelmann, E. D. Sontag. (1992) On the computational power of neural nets. *Proc. 5th ACM-Workshop on Computational Learning Theory,* 440-449.
